# Quantizing Density Estimators

**Peter Meinicke**
Neuroinformatics Group
University of Bielefeld
Bielefeld, Germany
*pmeinick@techfak.uni-bielefeld.de*

**Helge Ritter**
Neuroinformatics Group
University of Bielefeld
Bielefeld, Germany
*helge@techfak.uni-bielefeld.de*

## Abstract

We suggest a nonparametric framework for unsupervised learning of projection models in terms of density estimation on quantized sample spaces. The objective is not to optimally reconstruct the *data* but instead the quantizer is chosen to optimally reconstruct the *density* of the data. For the resulting *quantizing density estimator* (QDE) we present a general method for parameter estimation and model selection. We show how projection sets which correspond to traditional unsupervised methods like vector quantization or PCA appear in the new framework. For a principal component quantizer we present results on synthetic and real-world data, which show that the QDE can improve the generalization of the kernel density estimator although its estimate is based on significantly lower-dimensional projection indices of the data.

## 1 Introduction

Unsupervised learning is essentially concerned with finding alternative representations for unlabeled data. These alternative representations usually reflect some important properties of the underlying distribution and usually they try to exploit some redundancy in the data. In that way many unsupervised methods aim at a *complexity-reduced* representation of the data, like the most common approaches, namely vector quantization (VQ) and principal component analysis (PCA). Both approaches can be viewed as specific kinds of quantization, which is a basic mechanism of complexity reduction.

The objective of our approach to unsupervised learning is to achieve a suitable quantization of the data space which allows for an optimal reconstruction of the underlying density from a finite sample. In that way we consider unsupervised learning as density estimation on a quantized sample space and the resulting estimator will be referred to as *quantizing density estimator* (QDE). The construction of a QDE first requires to specify a suitable class of parametrized quantization functions and then to select from this set a certain function with good generalization properties. While the first point is common to unsupervised learning, the latter point is addressed in a density estimation framework where we tackle the model selection problem in a data-driven and nonparametric way.

It is often overlooked that modern Bayesian approaches to unsupervised learning and model selection are almost always based on some strong assumptions about the data distribution. Unfortunately these assumptions usually cannot be inferred from human knowledge about

the data domain and therefore the model building process is usually driven by computational considerations. Although our approach can be interpreted in terms of a generative model of the data, in contrast to most other generative models (see [10] for an overview), the present approach is nonparametric, since no specific assumptions about the functional form of the data distribution have to be made. In that way our approach compares well with other quantization methods, like principal curves and surfaces [4, 13, 6], which only have to make rather general assumptions about the underlying distribution. The QDE approach can utilize these methods as specific quantization techniques and shows a practical way how to further automatize the construction of unsupervised learning machines.

## 2 Quantization by Density Estimation

We will now explain how the QDE may be derived from a generalization of the kernel density estimator (KDE), one of the most popular methods for nonparametric density estimation [12, 11]. If we construct a kernel density estimator on the basis of a quantized sample, we have the following estimator

$$\hat{f}(\mathbf{y}) = \frac{1}{N} \sum_{i=1}^{N} K(\mathbf{y}, \mathbf{q}(\mathbf{y}_i; \boldsymbol{\theta})) \tag{1}$$

where $\{\mathbf{y}_1, \ldots, \mathbf{y}_N\}$ is a sample from the target distribution, $K(\cdot, \cdot)$ denotes the kernel function and $\mathbf{q} : \mathbb{R}^D \to \mathcal{P}_{\boldsymbol{\theta}} \subseteq \mathbb{R}^D$ with parameter vector $\boldsymbol{\theta}$ is a given quantization or projection function which maps a point $\mathbf{y}$ to a parametrized subset $\mathcal{P}_{\boldsymbol{\theta}}$ of the sample space

$$\mathbf{q}(\mathbf{y}; \boldsymbol{\theta}) = \mathbf{p}(\mathbf{s}_{\mathbf{p}}(\mathbf{y}); \boldsymbol{\theta}), \quad \mathbf{s}_{\mathbf{p}}(\mathbf{y}) = \arg\min_{\mathbf{x} \in \mathcal{X}} \|\mathbf{y} - \mathbf{p}(\mathbf{x}; \boldsymbol{\theta})\|. \tag{2}$$

Thereby the projection index $\mathbf{s}_{\mathbf{p}}(\cdot)$ associates a data point with its nearest neighbour in the projection set $\mathcal{P}_{\boldsymbol{\theta}}$ which is parametrized according to

$$\mathcal{P}_{\boldsymbol{\theta}} = \{\mathbf{y} : \mathbf{y} = \mathbf{p}(\mathbf{x}; \boldsymbol{\theta}), \mathbf{x} \in \mathcal{X} \subseteq \mathbb{R}^Q\} \tag{3}$$

where $\mathcal{X}$ is the set of all possible projection indices which are realizations of the deterministic latent variable $\mathbf{x}$. For a fixed non-zero kernel bandwidth the parameters of the quantization function may be determined by nonparametric maximum likelihood (ML) estimation, as will be introduced in the next section.

For an intuitive motivation of the QDE, one may ask from a data compression perspective whether it is necessary to store all the sample data $\{\mathbf{y}_1, \ldots, \mathbf{y}_N\}$ for the realization of the kernel density estimator or if it is possible to first reduce the data by some suitable quantization method and then construct the estimator from the more parsimonious complexity-reduced data set. Clearly, we would prefer a quantizer which does not decrease the performance of the estimator on unseen and unquantized data.

To get an idea of how to select a suitable quantization function let us consider an example from a 1D data space. In one dimension a natural projection set can be specified by a set of $M$ quantization levels on the real line, i.e. $\mathcal{P}_{\boldsymbol{\theta}} = \{w_1, \ldots, w_M\}$. For a fixed kernel bandwidth, we can now perform maximum likelihood estimation of the level coordinates. In that way we obtain a maximum likelihood estimator of the form

$$\hat{f}(y) = \frac{1}{N} \sum_{i=1}^{M} n_i K(y, \hat{w}_i) \tag{4}$$

with $n_i = |\{j : i = \arg\min_k |y_j - \hat{w}_k|\}|$ counting the number of data points which are quantized to level $i$. In this case, it remains the question how to choose the number of quantization levels.

From a different starting point the authors in [3] proposed the same functional form of a nonparametric ML density estimator with respect to Gaussian kernels of equal width centered on $N$ variable positions. As with the traditional Gaussian KDE (fixed kernel centers on $N$ data points), for consistency of the estimator the bandwidth has to be decreased as the sample size increases. In [3] the authors reported that for a fixed non-zero bandwidth, ML-estimation of the $N$ kernel centers always resulted in a smaller number of actually distinct centers, i.e. several kernels coincided to maximize the likelihood. Therefore the resulting estimator had the form of (4) where $M$ corresponds to the number of distinct centers with $n_i$ counting the number of kernels coinciding at $w_i$. The optimum number of effective quantization levels for a given bandwidth therefore arises as an automatic byproduct of ML estimation.

Finally one has to choose an appropriate kernel width which implicitly determines the complexity of the quantizer. The bandwidth selection problem has been tackled in the domain of kernel density estimation for some time and many approaches have been proposed (see e.g. [5] for an overview), among which the cross-validation methods are most common. In the next section we will adopt the method of likelihood cross-validation to find a practical answer to the bandwidth selection problem.

## 3   General Learning Scheme

By applying the method of sieves as proposed in [3], for a fixed non-zero bandwidth we can estimate the parameters of the quantization function via maximization of the log-likelihood $\sum_{i=1}^{N} \log(f(\mathbf{y}_i))$ w.r.t. to $\boldsymbol{\theta}$. For consistency of the resulting density estimator the bandwidth has to be decreased as the sample size increases, since asymptotically the estimator must converge to a mixture of delta functions centered on the data points. Thus, for decreasing bandwidth, the quantization function of the QDE must converge to the identity function, i.e. the QDE must converge to the kernel density estimator.

For a fixed bandwidth, maximization of the likelihood can be achieved by applying the EM-algorithm [2] which provides a convenient optimization scheme, especially for Gaussian kernels. The EM-scheme requires to iterate the following two steps

$$\textbf{E-Step:} \quad h_{ij}^{(m)} = K(\mathbf{y}_i, \mathbf{q}(\mathbf{y}_j; \boldsymbol{\theta}^{(m)})) / \sum_{k=1}^{N} K(\mathbf{y}_i, \mathbf{q}(\mathbf{y}_k; \boldsymbol{\theta}^{(m)})) \qquad (5)$$

$$\textbf{M-Step:} \quad \boldsymbol{\theta}^{(m+1)} = \arg \max_{\boldsymbol{\theta}} \sum_{i=1}^{N} \sum_{j=1}^{N} h_{ij}^{(m)} \log[K(\mathbf{y}_i, \mathbf{q}(\mathbf{y}_j; \boldsymbol{\theta}))] \qquad (6)$$

for a sequence $m = 0, 1, \ldots, n$ with suitable initial parameter vector $\boldsymbol{\theta}^{(0)}$ and sufficient convergence at $\boldsymbol{\theta}^{(n)}$. Thereby $h_{ij}$ denotes the posterior probability that data point $i$ has been "generated" by mixture component $j$ with density $K(\mathbf{y}, \mathbf{q}(\mathbf{y}_j; \boldsymbol{\theta}))$. For further insight one may realize that the M-Step requires to solve a constrained optimization problem by searching for

$$\max_{\boldsymbol{\theta}, \mathbf{X}} \sum_{i=1}^{N} \sum_{j=1}^{N} h_{ij} \log[K(\mathbf{y}_i, \mathbf{p}(\mathbf{x}_i; \boldsymbol{\theta}))], \quad \mathbf{X} = [\mathbf{x}_1, \ldots, \mathbf{x}_N] \qquad (7)$$

$$\text{subject to} \quad \mathbf{x}_i = \arg \min_{\mathbf{x} \in \mathcal{X}} \|\mathbf{y}_i - \mathbf{p}(\mathbf{x}; \boldsymbol{\theta})\|. \qquad (8)$$

In general this optimization problem can only be solved by iterative techniques. Therefore it may be convenient not to maximize but only to increase the log-likelihood at the M-Step which then corresponds to an application of the generalized EM-algorithm. Without (8)

unconstrained maximization according to (7) yields another class of interesting learning schemes which for reasons of space will not be considered in this paper.

For Gaussian kernels and an Euclidean metric for the projection, in the limiting case of a vanishing bandwidth, EM-optimization of the QDE parameters corresponds to minimization of the following error or risk

$$E_N(\boldsymbol{\theta}) = \frac{1}{N} \sum_{i=1}^{N} \min_{\mathbf{x} \in \mathcal{X}} \|\mathbf{y}_i - \mathbf{p}(\mathbf{x}; \boldsymbol{\theta})\|^2.$$

Minimization of such error functions corresponds to traditional approaches to unsupervised learning of projection models which can be viewed as a special case of QDE-based learning.

### 3.1 Bandwidth Selection

It is easy to see that the kernel bandwidth cannot be determined by ML-estimation since maximization of the likelihood would drive the bandwidth towards zero. For selection of the kernel bandwidth, we therefore apply the method of likelihood cross-validation (see e.g. [12]), which can be realized by a slight extension of the above EM-scheme. With the leave-one-out QDE

$$\hat{f}_{-i}(\mathbf{y}) = \frac{1}{N-1} \sum_{j \neq i} K(\mathbf{y}, \mathbf{q}(\mathbf{y}_j; \boldsymbol{\theta})) \tag{9}$$

the idea is to maximize $\sum_{i=1}^{N} \log(\hat{f}_{-i}(\mathbf{y}_i))$ with respect to the kernel bandwidth. For a Gaussian kernel with bandwidth $\sigma$ an appropriate EM scheme requires the following M-Step update rule

$$\hat{\sigma}^2 = \frac{1}{ND} \sum_{i=1}^{N} \sum_{j \neq i} h'_{ij} \|\mathbf{y}_i - \mathbf{q}(\mathbf{y}_j; \boldsymbol{\theta})\|^2 \tag{10}$$

The posterior probabilities $h'_{ij}$ are easily derived from a leave-one-out version of (5). In an overall optimization scheme one may now alter the estimation of $\boldsymbol{\theta}$ and $\sigma$ or alternatively one may estimate both by likelihood cross-validation.

## 4 Projection Sets in Multidimensions

By the specification of a certain class of quantization functions we can incorporate domain knowledge into the density estimation process, in order to improve generalization. Thereby the idea is to reduce the variance of the density estimator by reducing the variation of the quantized training set. The price is an increase of the bias which requires a careful selection of the set of admissible quantization functions. Then the QDE offers the chance to find a better bias/variance trade-off then with the "non-quantizing" KDE.

We will now show how to utilize existing methods for unsupervised learning within the current density estimation framework. Because many unsupervised methods can be stated in terms of finding optimal projection sets, it is straightforward to apply the corresponding types of quantization functions within the current framework. Thus in the following we shall consider specific parametrizations of the general projection set (3) which correspond to traditional unsupervised learning methods.

### 4.1 Vector Quantization

Vector quantization (VQ) is a standard technique among unsupervised methods and it is easily incorporated into the current density estimation framework by straightforward gen-

eralization of the one-dimensional quantizer in section 2 to the multi-dimensional case. Again with a fixed kernel bandwidth ML estimation yields a certain number of $M \leq N$ distinct ("effective") quantization levels, similar to maximum entropy clustering [9, 1].

The projection set of a vector quantizer can be parametrized according to a general basis function representation [7]

$$\mathbf{p}(x; \boldsymbol{\theta}) = \mathbf{W}\mathbf{b}(x), \quad x \in \{1, \ldots, N\} \tag{11}$$

with $N$-dimensional vector of basis functions $\mathbf{b}(\cdot)$ containing discrete delta functions, i.e. $b_j(k) = \delta_{jk}$ for component $j$.

The QDE on the basis of a vector quantizer can be expected to generalize well if some cluster structure is present within the data. In multi-dimensional spaces the data are often concentrated in certain regions which allows for a sparse representation by some reference vectors well-positioned in those regions. An alternative approach has been proposed in [14] where the application of the support vector formalism to density estimation results in a sparse representation of the data distribution.

## 4.2  Principal Component Analysis

A linear affine parametrization of the projection set yields candidate functions of the form

$$\mathbf{p}(\mathbf{x}; \boldsymbol{\theta}) = \mathbf{A}\mathbf{x} + \mathbf{c}, \quad \mathbf{x} \in \mathbb{R}^Q \tag{12}$$

with $Q \leq D$. The PCA approach reflects our knowledge that in most high-dimensional data spaces, the data are concentrated around some manifold of lower dimensionality.

To exploit this structure PCA divides the sample space into two subspaces which are quantized in different ways: within the "inner" subspace spanned by the directions of the projection manifold we have no quantization at all; within the orthogonal "outer" subspace the data are quantized to a single level.

With a Gaussian kernel with fixed bandwidth $\sigma$ the constrained optimization problem at the M-Step takes a convenient form which facilitates further analysis of the learning algorithm. From (7) and (8) it follows that one has to maximize the following objective function

$$L(\mathbf{c}, \mathbf{U}) = \text{const.} - \frac{1}{2\sigma^2} \sum_{i=1}^{N} \sum_{j=1}^{N} h_{ij} \|\mathbf{y}_i - \mathbf{U}\mathbf{U}^T(\mathbf{y}_j - \mathbf{c}) - \mathbf{c}\|^2 \tag{13}$$

where $D \times Q$ matrix $\mathbf{U}$ has orthogonal columns which span the subspace of the projection manifold. From the consideration of the corresponding stationarity conditions one finds that the sample mean $\bar{\mathbf{y}} = \frac{1}{N} \sum_i \mathbf{y}_i$ is an estimator of the shift vector $\mathbf{c}$.

Maximization of (13) with respect to $\mathbf{U}$ then requires to maximize the following trace

$$\text{tr}[\mathbf{U}^T(\mathbf{R} - \mathbf{S} - N\bar{\mathbf{y}}\bar{\mathbf{y}}^T)\mathbf{U}] = \text{tr}[\mathbf{U}^T \mathbf{Q} \, \mathbf{U}] \tag{14}$$

with symmetric matrices

$$\mathbf{R} = \sum_{i=1}^{N} \sum_{j=1}^{N} h_{ij}(\mathbf{y}_i\mathbf{y}_j^T + \mathbf{y}_j\mathbf{y}_i^T), \quad \mathbf{S} = \sum_{j=1}^{N} \mathbf{y}_j\mathbf{y}_j^T \sum_{i=1}^{N} h_{ij} \tag{15}$$

Thus (14) is maximized if $\mathbf{U}$ contains all eigenvectors of $\mathbf{Q}$, associated with positive eigenvalues, i.e. with $l_1, l_2, \ldots, l_d$ being the eigenvalues of $\mathbf{Q}$ we have the optimal subspace dimensionality

$$\hat{Q} = |\{l_i : l_i > 0, \ i = 1, \ldots, d\}| \tag{16}$$

which complements a recent result about parametric dimensionality estimation with respect to a $Q$-factor model with isotropic Gaussian noise [8]. For the QDE, the two limiting cases of zero and infinite bandwidth, are of particular interest. With the positive definite sample covariance matrix $\mathbf{C} = \frac{1}{N} \sum_i (\mathbf{y}_i - \bar{\mathbf{y}})(\mathbf{y}_i - \bar{\mathbf{y}})^T$ one can show

$$\lim_{\sigma \to \infty} \mathbf{Q} = -N\mathbf{C}, \quad \lim_{\sigma \to 0} \mathbf{Q} = N\mathbf{C}. \tag{17}$$

Thus for sufficiently large bandwidth $\mathbf{Q}$ becomes negative definite, which implies a zero subspace dimensionality estimator $\hat{Q} = 0$, i.e. all data are quantized to the sample mean. For sufficiently small bandwidth $\mathbf{Q}$ becomes positive definite implying $\hat{Q} = D$, i.e. no quantization takes place.

### 4.3 Independent Component Analysis

The PCA method provides a rather coarse quantization scheme since it only decides between one-level and no quantization for each subspace dimension. A natural refinement would therefore be to allow for a certain number of effective quantization levels for each component. Such an approach may be viewed as a nonparametric variant of independent component analysis (ICA). The idea is to quantize each coordinate axis separately, which yields a multi-dimensional quantization grid according to

$$\mathbf{p}(\mathbf{x}; \boldsymbol{\theta}) = \sum_{i=1}^{D} \mathbf{a}_i \mathbf{w}_i^T \mathbf{b}(x_i) + \mathbf{c}, \quad \mathbf{x} \in \{1, \ldots, N\}^D \tag{18}$$

with $\mathbf{a}_i \in \mathbb{R}^D$, $\mathbf{w}_i \in \mathbb{R}^N$ and $\mathbf{b}(\cdot)$ as in (11). Thereby the components of $\mathbf{w}_i$ contain the quantization levels of the $i$-th coordinate axis with direction $\mathbf{a}_i$. Further, it makes sense to normalize the direction vectors according to $\|\mathbf{a}_i\| = 1$. There are strong similarities with a parametric ICA model which has been suggested in [10], where source densities have been mixtures of delta functions and additive noise has been isotropic Gaussian.

Other unsupervised learning methods which correspond to different projection sets, like principal curves or multilayer perceptrons (see [7] for an overview) can as well be incorporated into the QDE framework and will be considered elsewhere.

## 5 Experiments

In the following experiments we investigated the PCA based QDE with Gaussian kernel and compared the generalization performance with that of the "non-quantizing" KDE. All parameters, including the bandwidth of the KDE, were estimated by likelihood cross-validation. In the first experiment we sampled 100 points from a stretched and rotated uniform distribution with support on a $1.0 \times 0.1$ rectangle. In this case the QDE extracted a one-dimensional "unquantized" subspace. Generalization performance was measured by the average log-likelihood on an independent 1000-point test set. With an automatically selected 1D subspace (compression ratio $2/1$) the PCA-QDE could improve the performance of the KDE from 1.96 to 2.04. Thus, the PCA-QDE could successfully exploit the elongated structure of the distribution. The estimated density functions are depicted in figure 1, where grey-values are proportional to $\hat{f}^{0.7}$ on a $128 \times 128$ grid. From the images one can see, that the QDE better captures the global structure of the distribution while the KDE is more sensitive to local variations in the data.

In a second experiment we trained PCA-QDEs with 64-dimensional real-world data ($8 \times 8$ images) which had been derived from the MNIST database of handwritten digits (http://www.research.att.com/~yann/ocr/mnist/). For each digit class a 1000-point training set and a 5000-point test set were used to compare the PCA-QDE with

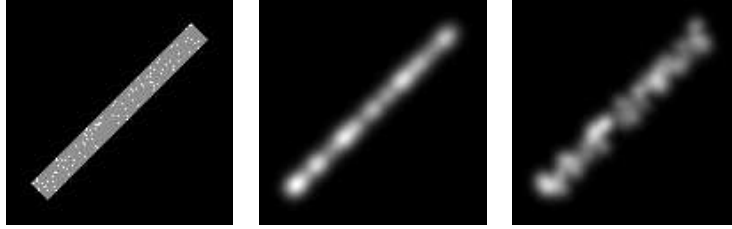

Figure 1: Left: stretched uniform distribution in 2D with white points indicating 100 data points used for estimation; middle: Estimated density using the PCA-QDE; right: kernel density estimate.

the KDE, with results shown in table 1. Again the PCA-QDE improved the generalization performance of the KDE although the QDE decided to remove about 40 "redundant" dimensions per digit class.

Table 1: Results on 64-dimensional digit data for different digit classes '0'...'9' (first row); second row: difference between average log-likelihoods of (PCA-)QDE and KDE on test set; third row: optimal subspace dimensionality of QDE

| Digit: | 0 | 1 | 2 | 3 | 4 | 5 | 6 | 7 | 8 | 9 |
|---|---|---|---|---|---|---|---|---|---|---|
| $\mathcal{L}_{QDE} - \mathcal{L}_{KDE}$: | 1.87 | 0.66 | 1.02 | 1.38 | 1.58 | 1.54 | 1.44 | 0.64 | 1.53 | 1.33 |
| $\hat{Q}$: | 22 | 29 | 26 | 24 | 24 | 25 | 24 | 27 | 21 | 25 |

## 6   Conclusion

The QDE offers a nonparametric approach to unsupervised learning of quantization functions which can be viewed as a generalization of the kernel density estimator. While the KDE is directly constructed from the given data set the QDE first creates a quantized representation of the data. Unlike traditional quantization methods which minimize the associated reconstruction error of the *data points*, the QDE adjusts the quantizer to optimize an estimate of the *data density*. This feature allows for a convenient model selection procedure, since the complexity of the quantizer can be controlled by the kernel bandwidth, which in turn can be selected in a data-driven way. For a practical realization we outlined EM-schemes for parameter estimation and bandwidth selection. As an illustration, we discussed examples with different projection sets which correspond to VQ, PCA and ICA methods. We presented experiments which demonstrate that the bias imposed by the quantization can lead to an improved generalization as compared to the "non-quantizing" KDE. This suggests that QDEs offer a promising approach to unsupervised learning that allows to control bias without the usually rather strong distributional assumptions of the Bayesian approach.

## Acknowledgement

This work was funded by the Deutsche Forschungsgemeinschaft within the project SFB 360.

# References

[1] J. M. Buhmann and N. Tishby. Empirical risk approximation: A statistical learning theory of data clustering. In C. M. Bishop, editor, *Neural Networks and Machine Learning*, pages 57–68. Springer, Berlin Heidelberg New York, 1998.

[2] A. P. Dempster, N. M. Laird, and D. B. Rubin. Maximum likelihood from incomplete data via the EM algorithm. *Journal of the Royal Statistical Society Series B*, 39:1–38, 1977.

[3] Stuart Geman and Chii-Ruey Hwang. Nonparametric maximum likelihood estimation by the method of sieves. *The Annals of Statistics*, 10(2):401–414, 1982.

[4] T. Hastie and W. Stuetzle. Principal curves. *Journal of the American Statistical Association*, 84:502–516, 1989.

[5] M. C. Jones, J. S. Marron, and S. J. Sheather. A brief survey of bandwidth selection for density estimation. *Journal of the American Statistical Association*, 91(433):401–407, 1996.

[6] B. Kégl, A. Krzyzak, T. Linder, and K. Zeger. Learning and design of principal curves. *IEEE Transaction on Pattern Analysis and Machine Intelligence*, 22(3):281–297, 2000.

[7] Peter Meinicke. *Unsupervised Learning in a Generalized Regression Framework*. PhD thesis, Universitaet Bielefeld, 2000. `http://archiv.ub.uni-bielefeld.de/disshabi/2000/0033/`.

[8] Peter Meinicke and Helge Ritter. Resolution-based complexity control for Gaussian mixture models. *Neural Computation*, 13(2):453–475, 2001.

[9] K. Rose, E. Gurewitz, and G. C. Fox. Statistical mechanics and phase transitions in clustering. *Physical Review Letters*, 65(8):945–948, 1990.

[10] Sam Roweis and Zoubin Ghahramani. A unifying review of linear Gaussian models. *Neural Computation*, 11(2):305–345, 1999.

[11] D. W. Scott. *Multivariate Density Estimation*. Wiley, 1992.

[12] B. W. Silverman. *Density Estimation for Statistics and Data Analysis*. Chapman and Hall, London and New York, 1986.

[13] Alex J. Smola, Robert C. Williamson, Sebastian Mika, and Bernhard Schölkopf. Regularized principal manifolds. In *Proc. 4th European Conference on Computational Learning Theory*, volume 1572, pages 214–229. Springer-Verlag, 1999.

[14] Vladimir N. Vapnik and Sayan Mukherjee. Support vector method for multivariate density estimation. In S. A. Solla, T. K. Leen, and K.-R. Müller, editors, *Advances in Neural Information Processing Systems*, volume 12, pages 659–665. The MIT Press, 2000.
